# Clustering with the Connectivity Kernel

**Bernd Fischer, Volker Roth and Joachim M. Buhmann**
Institute of Computational Science
Swiss Federal Institute of Technology Zurich
CH-8092 Zurich, Switzerland
{bernd.fischer, volker.roth,jbuhmann}@inf.ethz.ch

## Abstract

Clustering aims at extracting hidden structure in dataset. While the problem of finding *compact* clusters has been widely studied in the literature, extracting arbitrarily formed *elongated* structures is considered a much harder problem. In this paper we present a novel clustering algorithm which tackles the problem by a two step procedure: first the data are transformed in such a way that elongated structures become compact ones. In a second step, these new objects are clustered by optimizing a compactness-based criterion. The advantages of the method over related approaches are threefold: (i) robustness properties of compactness-based criteria naturally transfer to the problem of extracting elongated structures, leading to a model which is highly robust against outlier objects; (ii) the transformed distances induce a Mercer kernel which allows us to formulate a polynomial approximation scheme to the generally $\mathcal{NP}$-hard clustering problem; (iii) the new method does not contain free kernel parameters in contrast to methods like spectral clustering or mean-shift clustering.

## 1 Introduction

Clustering or grouping data is an important topic in machine learning and pattern recognition research. Among various possible grouping principles, those methods which try to find *compact* clusters have gained particular importance. Presumably the most prominent method of this kind is the $K$-means clustering for vectorial data [6]. Despite the powerful modeling capabilities of compactness-based clustering methods, they mostly fail in finding *elongated structures*. The fast single linkage algorithm [9] is the most often used algorithm to search for elongated structures, but it is known to be very sensitive to outliers in the dataset. Mean shift clustering [3], another method of this class, is capable of extracting elongated clusters only if all modes of the underlying probability distribution have one single maximum. Furthermore, a suitable kernel bandwidth parameter has to be preselected [2]. Spectral clustering [10] shows good performance in many cases, but the algorithm is only analyzed for special input instances while a complete analysis of the algorithm is still missing. Concerning the preselection of a suitable kernel width, spectral clustering suffers from similar problems as mean shift clustering.

In this paper we present an alternative method for clustering elongated structures. Apart from the number of clusters, it is a completely parameter-free grouping principle. We build up on the work on *path-based clustering* [7]. For a slight modification of the original prob-

lem we show that the defined path distance induces a kernel matrix fulfilling Mercers condition. After the computation of the path-based distance, the compactness-based pairwise clustering principle is used to partition the data. While for the general $\mathcal{NP}$-hard pairwise clustering problem no approximation algorithms are known, we present a polynomial time approximation scheme (PTAS) for our special case with path-based distances. The Mercer property of these distances allows us to embed the data in a $(n-1)$ dimensional vector space even for non-metric input graphs. In this vector space, pairwise clustering reduces to minimizing the $K$-means cost function in $(n-1)$ dimensions [13]. For the latter problem, however, there exists a PTAS [11].

In addition to this theoretical result, we also present an efficient practical algorithm resorting to a 2-approximation algorithm which is based on kernel PCA. Our experiments suggest that kernel PCA effectively reduces the noise in the data while preserving the coarse cluster structure. Our method is compared to spectral clustering and mean shift clustering on selected artificial datasets. In addition, the performance is demonstrated on the USPS handwritten digits dataset.

## 2 Clustering by Connectivity

The main idea of our clustering criterion is to transform elongated structures into compact ones in a preprocessing step. Given the transformed data, we then infer a clustering solution by optimizing a compactness based criterion. The advantage of circumventing the problem of directly finding connected (elongated) regions in the data as e.g. in the spanning tree approach is the following: while spanning tree algorithms are extremely sensitive to outliers, the two-step procedure may benefit from the statistical robustness of certain compactness based methods. Concerning the general case of datasets which are not given in a vector space, but only characterized by pairwise dissimilarities, the *pairwise clustering* model has been shown to be robust against outliers in the dataset [12]. It may, thus, be a natural choice to formulate the second step as searching for the partition vector $c \in \{1, \ldots, K\}^n$ that minimizes the pairwise clustering cost function

$$H^{\text{PC}}(c; D) = \sum_{\nu=1}^{K} \frac{1}{n_\nu} \sum_{i:c_i=\nu} \sum_{j:c_j=\nu} d_{ij}, \tag{1}$$

where $K$ denotes the number of clusters, $n_\nu = |\{i : c_i = \nu\}|$ denotes the number of objects in cluster $\nu$, and $d_{ij}$ is the pairwise "effective" dissimilarity between objects $i$ and $j$ as computed by a preprocessing step.

The idea of this preprocessing step is to define distances between objects by considering certain paths through the total object set. The natural formalization of such path problems is to represent the objects as a graph: consider a connected graph $G = (V, E, d')$ with $n$ vertices (the objects) and *symmetric nonnegative* edge weights $d'_{ij}$ on the edge $(i, j)$ (the original dissimilarities). Let us denote by $\mathcal{P}_{ij}$ all paths from vertex $i$ to vertex $j$. In order to make those objects more similar which are connected by "bridges" of other objects, we define for each path $p \in \mathcal{P}_{ij}$ the *effective dissimilarity* $d_{ij}^p$ between $i$ and $j$ connected by $p$ as the maximum weight on this path, i.e. the "weakest link" on this path. The total dissimilarity between vertices $i$ and $j$ is then defined as the minimum of all path-specific effective dissimilarities $d_{ij}^p$:

$$d_{ij} := \min_{p \in \mathcal{P}_{ij}} \left\{ \max_{1 \le h \le |p|-1} d'_{p[h]p[h+1]} \right\}. \tag{2}$$

Figure 1 illustrates the definition of the effective dissimilarity. If the objects are in the same cluster their pairwise effective dissimilarities will be small (fig. 1(a)). If the two objects belong to two different clusters, however, all paths contain at least one large dissimilarity and the resulting effective dissimilarity will be large (fig. 1(b)). Note that single outliers as in (fig. 1(a,b)) do not affect the basic structure in the path-based distances. A problem

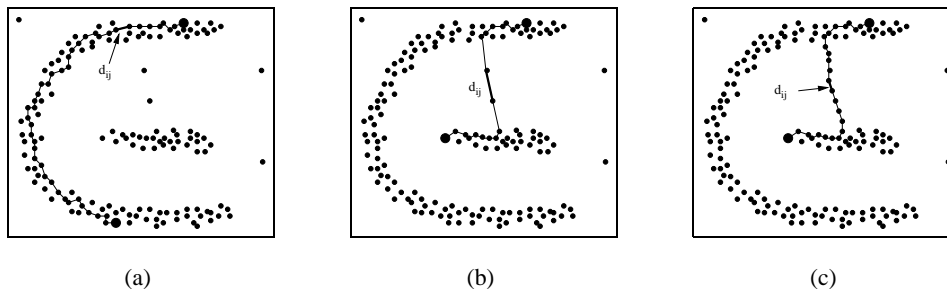

Figure 1: Effective dissimilarities. (a) If objects belong to the same high-density region, $d_{ij}$ is small. (b) If they are in different regions, $d_{ij}$ is larger. (c) To regions connected by a "bridge".

can only occur, if the point density along a "bridge" between the two clusters is as high as the density on the backbone of the clusters, see 1(c). In such a case, however, the points belonging to the "bridge" can hardly be considered as "outliers". The reader should notice that the single linkage algorithm does not posses the robustness properties, since it will separate the three most distant outlier objects in example 1(a) from the remaining data, but it will not detect the dominant structure.

Summarizing the above model, we formalize the **path-based clustering problem** as:
**INPUT:** A symmetric $(n \times n)$ matrix $D' = (d'_{ij})_{1 \le i,j \le n}$ of nonnegative pairwise dissimilarities between $n$ objects, with zero diagonal elements.
**QUESTION:** Find clusters by minimizing $H^{\mathrm{PC}}(c; D)$, where the matrix $D$ represents the effective dissimilarities derived from $D'$ by eq. (2).

## 3  The Connectivity Kernel

In this section we show that the effective dissimilarities induce a Mercer kernel on the weighted graph $G$. The Mercer property will then allow us to derive several approximation results for the $\mathcal{NP}$-hard pairwise clustering problem in section 4.

**Definition 1.** A metric $D$ is called *ultra-metric* if it satisfies the condition $d_{ij} \le \max(d_{ik}, d_{kj})$ for all distinct $i, j, k$.

**Theorem 1.** *The dissimilarities defined by (2) induce an ultra-metric on $G$.*

*Proof.* We have to check the axioms of a metric distance measure plus the restricted triangle inequality $d_{ij} \le \max(d_{ik}, d_{kj})$: (i) $d_{ij} \ge 0$, since the weights are nonnegative; (ii) $d_{ij} = d_{ji}$, since we consider symmetric weights; (iii) $d_{ii} = 0$ follows immediately from definition (2); (iv) The proof of the restricted triangle inequality follows by contradiction: suppose, there exists a triple $i, j, k$ for which $d_{ij} > \max(d_{ik}, d_{kj})$. This situation, however, contradicts the above definition (2) of $d_{ij}$: in this case there exists a path from $i$ to $j$ over $k$, the weakest link of which is shorter than $d_{ij}$. Equation (2) then implies that $d_{ij}$ must be smaller or equal to $\max(d_{ik}, d_{kj})$. $\qquad\square$

**Definition 2.** A metric $D$ is $\ell_2$ embeddable, if there exists a set of vectors $\{x_i\}_{i=1}^n$, $x_i \in \mathbb{R}^p$, $p \le n - 1$ such that for all pairs $i, j$ $\|x_i - x_j\|_2 = d_{ij}$.
A proof for the following lemma has been given in [4]:

**Lemma 1.** *For every ultra-metric $D$, $\sqrt{D}$ is $\ell_2$ embeddable.*
Now we are considered with a realization of such an embedding. We introduce the notion of a *centralized matrix*. Let $P$ be an $(n \times n)$ matrix and let $Q = I_n - \frac{1}{n} e_n e_n^\top$, where $e_n = (1, 1, \ldots 1)^\top$ is a $n$-vector of ones and $I_n$ the $n \times n$ identity matrix. We define the *centralized $P$* as $P^c = QPQ$.
The following lemma (for a proof see e.g. [15]) characterizes $\ell_2$ embeddings:

**Lemma 2.** *Given a metric $D$, $\sqrt{D}$ is $\ell_2$ embeddable iff $D^c = QDQ$ is negative (semi)definite.*

The combination of both lemmata yields the following theorem.

**Theorem 2.** *For the distance matrix $D$ defined in the setting of theorem 1, the matrix $S^c = -\frac{1}{2}D^c$ with $D^c = QDQ$ is a Gram matrix or Mercer kernel. It contains dot products between a set of vectors $\{x_i\}_{i=1}^n$ with squared Euclidean distances $\|x_i - x_j\|_2^2 = d_{ij}$.*

*Proof.* (i) Since $D$ is ultra-metric, $\sqrt{D}$ is $\ell_2$ embeddable by lemma 1, and $D^c$ is negative (semi)definite by lemma (2). Thus, $S^c = -\frac{1}{2}D^c$ is positive (semi)definite. As any positive (semi)definite matrix, $S^c$ defines a Gram matrix or Mercer kernel. (ii) Since $s_{ij}^c$ is a dot-product between two vectors $x_i$ and $x_j$, the squared Euclidean distance between $x_i$ and $x_j$ is defined by $\|x_i - x_j\|_2^2 = s_{ii}^c + s_{jj}^c - 2s_{ij}^c = -\frac{1}{2}\left[d_{ii}^c + d_{jj}^c - 2d_{ij}^c\right]$. With the definition of the centralized distances, it can be seen easily that all but one term, namely the original distance, cancel out: $-\frac{1}{2}\left[d_{ii}^c + d_{jj}^c - 2d_{ij}^c\right] = d_{ij}$. □

## 4    Approximation Results

Pairwise clustering is known to be $\mathcal{NP}$-hard [1]. To our knowledge there is no polynomial time approximation algorithm known for the general case of pairwise clustering. For our special case in which the data are transformed into *effective dissimilarities*, however, we now present a polynomial time approximation scheme.

**A Polynomial Time Approximation Scheme.** Let us first consider the computation of the effective dissimilarities $D$. Despite the fact that the path-based distance is a minimum over all paths from $i$ to $j$, the whole distance matrix can be computed in polynomial time.

**Lemma 3.** *The path-based dissimilarity matrix $D$ defined by equation 2 can be computed in running time $\mathcal{O}(n^2 \log n)$.*

*Proof.* The computation of the connectivity kernel matrix is an extention of Kruskal's minimum spanning tree algorithm. We start with $n$ clusters each containing one single object. In each iteration step the two clusters $C_i$ and $C_j$ are merged with minimal costs $d_{ij} = \min_{p \in C_i, q \in C_j} d'_{pq}$ where $d'_{pq}$ is the edge weight on the input graph. The link $d_{ij}$ gives the effective dissimilarity of all objects in $C_i$ to all objects in $C_j$. To proof this, one can consider the case, where $d_{ij}$ is not the effective dissimilarity between $C_i$ and $C_j$. Then there exists a path over some other cluster $C_k$, where all objects on this path have a smaller weight, implying the existence of another pair of clusters with smaller merging costs. The running time is $\mathcal{O}(n^2 \log n)$ for the spanning tree algorithm on the the complete input graph and additional $\mathcal{O}(n^2)$ for filling all elements in the matrix $D$. □

Let us now discuss the clustering step. Recall first the problem of $K$-means clustering: given $n$ vectors $\mathcal{X} = \{x_1, \ldots, x_n \in \mathbb{R}^p\}$, the task is to partition the vectors in such a way that the squared Euclidean distance to the cluster centroids is minimized. The objective function for $K$-means is given by

$$H^{\mathrm{KM}}(c; \mathcal{X}) = \sum_{\nu=1}^K \sum_{i:c_i=\nu}(x_i - y_\nu)^2 \qquad \text{where} \qquad y_\nu = \frac{1}{n_\nu}\sum_{j:c_j=\nu} x_j \quad (3)$$

Minimizing the $K$-means objective function for squared Euclidean distances is $\mathcal{NP}$-hard if the dimension of the vectors is growing with $n$.

**Lemma 4.** *There exists a polynomial time approximation scheme (PTAS) for $H^{KM}$ in arbitrary dimensions and for fixed $K$.*

*Proof.* In [11] Ostrovsky and Rabani presented a PTAS for $K$-means. □

Using this approximation lemma we are able to proof the existence of a PTAS for pairwise data clustering using the distance defined by (2).

**Theorem 3.** *for distances defined by (2), there exists a PTAS for $H^{PC}$.*

*Proof.* By lemma 3 the dissimilarity matrix $D$ can be computed in polynomial time. By theorem 2 we can find vectors $x_1, \ldots x_n \in \mathbb{R}^p$ ($p \leq n - 1$) with $d_{ij} = ||x_i - x_j||_2^2$. For squared Euclidean distances, however, there is an algebraic identity between $H^{PC}(c; D)$ and $H^{KM}(c; \mathcal{X})$ [13]. By lemma 4 there exists a PTAS for $H^{KM}$ and thus for $H^{PC}$. $\square$

**A 2-approximation by Kernel PCA.** While the existence of a PTAS is an interesting theoretical approximation result, it does not automatically follow that a PTAS can be used in a constructive way to derive practical algorithms. Taking such a practical viewpoint, we now consider another (weaker) approximation result from which, however, an efficient algorithm can be designed easily. From the fact that we can define a connectivity kernel matrix we can use kernel PCA [14] to reduce the data dimension. The vectors are projected on the first principle components. Diagonalization of the centered kernel matrix $S^c$ leads to $S^c = V^t \Lambda V$, with an orthogonal matrix $V = (v_1, \ldots, v_n)$ containing the eigenvectors of $S^c$, and a diagonal matrix $\Lambda = diag(\lambda_1, \ldots, \lambda_n)$ containing the corresponding eigenvalues on its diagonal. Assuming now that the eigenvalues are in descending order ($\lambda_1 \geq \lambda_2 \geq \cdots \geq \lambda_n$), the data are projected on the first $p$ eigenvectors: $x_i' = \sum_{j=1}^{p} \sqrt{\lambda_j} v_{ji}$.

**Theorem 4.** *Embedding the path-based distances into $\mathbb{R}^K$ by kernel PCA and enumerating over all possible Voronoi partitions yields an $\mathcal{O}(n^{K^2+1})$ algorithm which approximates path-based clustering within a constant factor of 2.*

*Proof.* The solution of the $K$-means cost function induces a Voronoi partition on the dataset. If the dimension $p$ of the data is kept fix, the number of different Voronoi partitions is at most $\mathcal{O}(n^{Kp})$, and they can be enumerated in $\mathcal{O}(n^{Kp+1})$ time [8]. Further, if the embedding dimension is chosen as $p = K$, $K$-means in $\mathbb{R}^K$ is a 2-approximation algorithm for $K$-means in $\mathbb{R}^{n-1}$ [5]. Combining both results, we arrive at a 2-approximation algorithm with running time $\mathcal{O}(n^{K^2+1})$. $\square$

**Heuristics without approximation guarantees.** The running time of the 2-approximation algorithm may still be too large for many applications, therefore we will refer to two heuristic optimization methods without approximation guarantees. Instead of enumerating all possible Voronoi partitions, one can simply partition the data with the fast classical $K$-means algorithm. In one sweep it assigns each object to the nearest centroid, while keeping all other object assignments fixed. Then the centroids are relocated according to the new assignments. Since the running time grows linear with the data dimension, it is useful to first embed the data in $K$ dimensions which leads us to a functional which optimal solution is even in the worst case within a factor of two of the desired solution, as we know from the above approximation results. In this reduced space, the $K$-means heuristics is applied with the hope that there exist only few local minima in the low-dimensional subspace.
As a second heuristic one can apply Ward's method which is an agglomerative optimization of the $K$-means objective function.[1] It starts with $n$ clusters, each containing one object, and in each step the two clusters that minimize the $K$-means objective function are merged. Ward's method produces a cluster hierarchy. For applications of this method see figure 3.

## 5   Experiments

We first compare our method with the classical single linkage algorithm on artificial data consisting of three noisy spirals, see figure 2. Our main concern in these experiments is the robustness against noise in the data. Figure 3(a) shows the dendrogram produced by single linkage. The leaves of the tree are the objects of figure 2. For better visualization of the tree structure, the bar diagrams below the tree show the labels of the three cluster

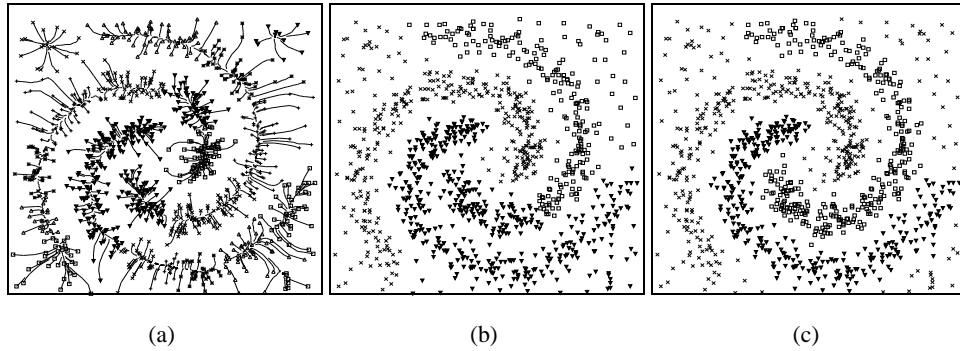

| (a) | (b) | (c) |

Figure 2: Comparison to other clustering methods. (a) Mean shift clustering, (b) Spectral Clustering, (c) Connectivity kernel clustering. (Color images at http://www.inf.ethz.ch/~befische/nips03)

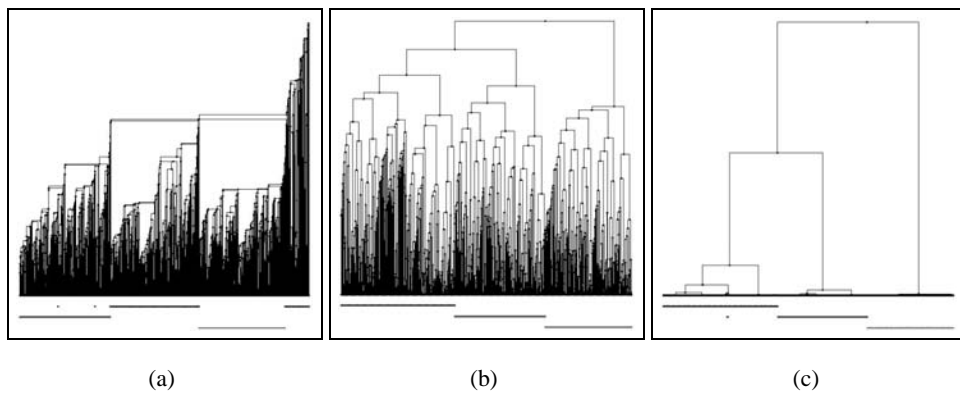

| (a) | (b) | (c) |

Figure 3: Hierarchical Clustering Solutions for example 2(c). (a) Single Linkage, (b) Ward's method with connectivity kernel, applied to embedded objects in $n - 1$ dimensions. (c) Ward's method after kernel PCA embedding in 3 dimensions.

solution as drawn in fig. 2(c). The height of the inner nodes depicts the merging costs for two subtrees. Each level of the hierarchy is one cluster solution. It is obvious that the main parts of the spiral arms are found, but the objects drawn on the right side are separated from the rest of the cluster. The respective objects are the outliers that are separated in the highest hierarchical levels of the algorithm. We conclude that for small $K$, single linkage has the tendency to separates single outlier objects from the data.

By way of the connectivity kernel we can transform the original dyadic data to $n - 1$ dimensional vectorial data. To show comparable results for the connectivity kernel, we apply Ward's method to the embedded vectors. Figure 3(b) shows the cluster hierarchy for Ward's method in the full space of $n - 1$ dimensions. Opposed to the single linkage results, the main structure of the spiral arms has been successfully found in the hierarchy corresponding to the three cluster solution. Below the three cluster lever, the tree appears to be very noisy. It should also be noticed that the costs of the three cluster solution are not much larger as the costs of the four cluster solution, indicating that the three cluster solution does not form a distinctly separated hierarchical level.

Figure 3(c) demonstrates that more distinctly separated levels can be found after applying kernel PCA and embedding the objects into a low-dimensional space (here 3 dimensions). Ward's method is then applied to the embedded objects. One can see that the coarse struc-

ture of the tree has been preserved, while the costs of cluster solutions for $K > 3$ have been shrunken towards zero. We conclude that PCA has the effect of de-noising the hierarchical tree, leading to a more robust agglomerative algorithm.

Now we compare our results to other recently published clustering techniques, that have been designed to extract elongated structures. Mean shift clustering [3] computes a trajectory of vectors towards the gradient of the underlying probability density. The probability distribution is estimated with a density estimation kernel, e.g. a Gaussian kernel. The trajectories starting at each point in the feature space converge at the local maxima of the probability distribution. Mean shift clustering is only applicable to finite dimensional vector spaces, because it implicitly involves density estimation. A potential shortcoming of mean-shift clustering is the following: if the modes of the distribution have multiple local maxima (as e.g. in the spiral arm example), there does not exist any kernel bandwidth to successfully separate the data according to the underlying structure. In figure 2(a) the best result for mean shift clustering is drawn. For smaller values of $\sigma$ the spiral arms are further subdivided into additional clusters, and for a larger bandwidth values, the result becomes more and more similar to compactness-based criteria like $K$-means.

Spectral methods [10] have become quite popular in the last years. Usually the Laplacian matrix based on a Gaussian kernel is computed. By way of PCA, the data are embedded in a low dimensional space. The $K$-means algorithm on the embedded data then gives the resulting partition. It has also been proposed to project the data on the unit sphere before applying $K$-means. Spectral clustering with a Gaussian kernel is known to be able to separate nested circles, but we observed that it has severe problems to extract the noisy spiral arms, see 2(b). In spectral clustering, the kernel width $\sigma$ is a free parameter which has to be selected "correctly". If $\sigma$ is too large, spectral clustering becomes similar to standard $K$-means and fails to extract elongated structures. If, on the other hand, $\sigma$ is too small, the algorithm becomes increasingly sensitive to outliers, in the sense that it has the tendency to separate single outlier objects.

Our approach to clustering with the connectivity kernel, however, could successfully extract the three spiral arms as can be seen in figure 2(c). The reader should notice, that this method does not require the user to preselect any kernel parameter.

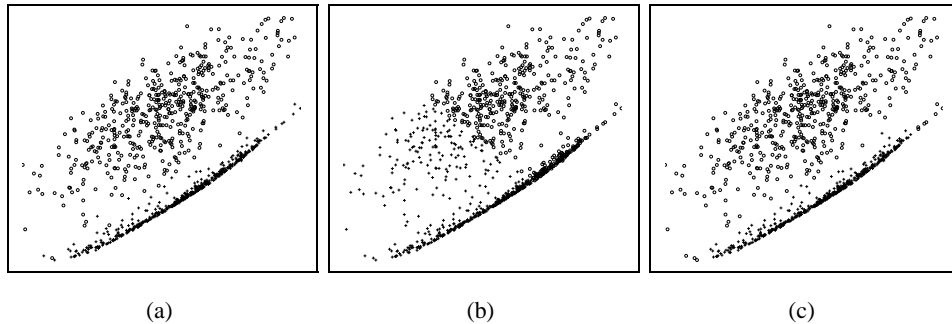

        (a)                   (b)                 (c)

Figure 4: Example from the USPS dataset. Training example of digits 2 and 9 embedded in two dimensions. (a) Ground truth labels. (b) $K$-means labels and (c) clustering with connectivity kernel.

In a last experiment, we show the advantages of our method compared to a parameter-free compactness criterion ($K$-means) on the problem of clustering digits '2' and '9' from the USPS digits dataset. Figure 4 shows the clustering result of our method using the connectivity kernel. The 16x16 digit gray-value images of the USPS dataset are interpreted as vectors and projected on the two leading principle components. In figure 4(a) the ground truth solution is drawn. Figure 4(b) shows the partition by directly applying $K$-means clustering, and figure 4(c) shows the result produced by our method. Compared to the ground

truth solution, path-based clustering succeeded in extracting the elongated structures, resulting in a very small error of only $1.5\%$ mislabeled digits. The compactness-based $K$-means method, on the other hand, produces clearly suboptimal clusters with an error rate of $30.6\%$.

## 6   Conclusion

In this paper we presented a clustering approach, that is based on path-based distances in the input graph. In a first step, elongated structures are transformed into compact ones, which in the second step are partitioned by the compactness-based pairwise clustering method. We showed that the transformed distances induce a Mercer kernel, which in turn allowed us to derive a polynomial time approximation scheme for the generally $\mathcal{NP}$-hard pairwise clustering problem. Moreover, Mercers property renders it possible to embed the data into low-dimensional subspaces by Kernel PCA. These embeddings form the basis for an efficient 2-approximation algorithm, and also for de-noising the data to "robustify" fast agglomerative optimization heuristics. Compared to related methods like single linkage, mean shift clustering and spectral clustering, our method has been shown to successfully overcome the problem of sensitivity to outlier objects, while being capable of extracting nested elongated structures. Our method does not involve any free kernel parameters, which we consider to be a particular advantage over both mean shift– and spectral clustering.

## Footnotes

[1]It has been shown in [12] that Ward's method is an optimization heuristics for $H^{PC}$. Due to the equivalence of $H^{PC}$ and $H^{KM}$ in our special case, this property carries over to $K$-means.

## References

[1]  P. Brucker. On the complexity of clustering problems. *Optimization and Operations Research*, pages 45–54, 1977.

[2]  D. Comaniciu. An algorithm for data-driven bandwidth selection. *IEEE T-PAMI*, 25(2):281–288, 2003.

[3]  D. Comaniciu and P. Meer. Mean shift: A robust approach toward feature space analysis. *IEEE T-PAMI*, 24(5):603–619, 2002.

[4]  M. Deza and M. Laurent. Applications of cut polyhedra. *J. Comp. Appl. Math.*, 55:191–247, 1994.

[5]  P. Drineas, A. Frieze, R. Kannan, S. Vempala, and V. Vinay. Clustering on large graphs and matrices. In *Proc. of the ACM-SIAM Symp. on Discrete Algorithm.*, pages 291–299, 1999.

[6]  R. Duda, P. Hart, and D. Stork. *Pattern Classification*. Wiley & Sons, 2001.

[7]  B. Fischer and J.M. Buhmann. Path-based clustering for grouping of smooth curves and texture segmentation. *IEEE T-PAMI*, 25(4):513–518, 2003.

[8]  M. Inaba, N. Katoh, and H. Imai. Applications of weighted voronoi diagrams and randomization to variance-based $k$-clustering. In *10th ACM Sympos. Computat. Geom.*, pages 332–339, 1994.

[9]  A. Jain and R. Dubes. *Algorithms for Clustering Data*. Prentice Hall, 1988.

[10]  A.Y. Ng, M.I. Jordan, and Y. Weiss. On spectral clustering: Analysis and an algorithm. In *NIPS*, volume 14, pages 849–856, 2002.

[11]  R. Ostrovsky and Y. Rabani. Polynomial time approximation schemes for geometric min-sum median clustering. *Journal of the ACM*, 49(2):139–156, 2002.

[12]  J. Puzicha, T. Hofmann, and J.M. Buhmann. A theory of proximity based clustering: Structure detection by optimization. *Pattern Recognition*, 2000.

[13]  V. Roth, J. Laub, J.M. Buhmann, and K.-R. Müller. Going metric: Denoising pairwise data. In *NIPS*, volume 15, 2003. to appear.

[14]  B. Schölkopf, A. Smola, and K.-R. Müller. Nonlinear component analysis as a kernel eigenvalue problem. *Neural Computation*, 10:1299–1319, 1998.

[15]  G. Young and A. S. Householder. Discussion of a set of points in terms of their mutual distances. *Psychometrica*, 3:19–22, 1938.
